# Distributed Occlusion Reasoning for Tracking with Nonparametric Belief Propagation

**Erik B. Sudderth, Michael I. Mandel, William T. Freeman, and Alan S. Willsky**
Department of Electrical Engineering and Computer Science
Massachusetts Institute of Technology
*esuddert@mit.edu, mim@alum.mit.edu, billf@mit.edu, willsky@mit.edu*

## Abstract

We describe a three–dimensional geometric hand model suitable for visual tracking applications. The kinematic constraints implied by the model's joints have a probabilistic structure which is well described by a graphical model. Inference in this model is complicated by the hand's many degrees of freedom, as well as multimodal likelihoods caused by ambiguous image measurements. We use nonparametric belief propagation (NBP) to develop a tracking algorithm which exploits the graph's structure to control complexity, while avoiding costly discretization.

While kinematic constraints naturally have a local structure, self–occlusions created by the imaging process lead to complex interpendencies in color and edge–based likelihood functions. However, we show that local structure may be recovered by introducing binary hidden variables describing the occlusion state of each pixel. We augment the NBP algorithm to infer these occlusion variables in a distributed fashion, and then analytically marginalize over them to produce hand position estimates which properly account for occlusion events. We provide simulations showing that NBP may be used to refine inaccurate model initializations, as well as track hand motion through extended image sequences.

## 1 Introduction

Accurate visual detection and tracking of three–dimensional articulated objects is a challenging problem with applications in human–computer interfaces, motion capture, and scene understanding [1]. In this paper, we develop a probabilistic method for tracking a geometric hand model from monocular image sequences. Because articulated hand models have many (roughly 26) degrees of freedom, exact representation of the posterior distribution over model configurations is intractable. Trackers based on extended and unscented Kalman filters [2, 3] have difficulties with the multimodal uncertainties produced by ambiguous image evidence. This has motived many researchers to consider nonparametric representations, including particle filters [4, 5] and deterministic multiscale discretizations [6]. However, the hand's high dimensionality can cause these trackers to suffer catastrophic failures, requiring the use of models which limit the hand's motion [4] or sophisticated prior models of hand configurations and dynamics [5, 6].

An alternative way to address the high dimensionality of articulated tracking problems is to describe the posterior distribution's statistical structure using a *graphical model*. Graph-

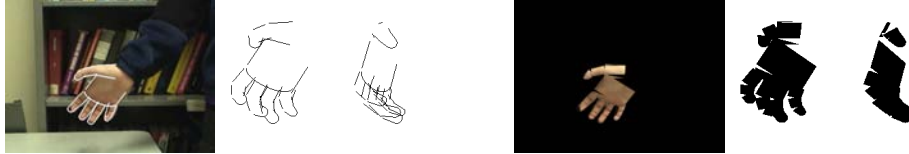

Figure 1: Projected edges (left block) and silhouettes (right block) for a configuration of the 3D structural hand model matching the given image. To aid visualization, the model is also projected following rotations by $35°$ (center) and $70°$ (right) about the vertical axis.

ical models have been used to track view–based human body representations [7], contour models of restricted hand configurations [8], view–based 2.5D "cardboard" models of hands and people [9], and a full 3D kinematic human body model [10]. Because the variables in these graphical models are continuous, and discretization is intractable for three–dimensional models, most traditional graphical inference algorithms are inapplicable. Instead, these trackers are based on recently proposed extensions of particle filters to general graphs: mean field Monte Carlo in [9], and *nonparametric belief propagation* (NBP) [11, 12] in [10].

In this paper, we show that NBP may be used to track a three–dimensional geometric model of the hand. To derive a graphical model for the tracking problem, we consider a redundant *local* representation in which each hand component is described by its own three–dimensional position and orientation. We show that the model's kinematic constraints, including self–intersection constraints not captured by joint angle representations, take a simple form in this local representation. We also provide a local decomposition of the likelihood function which properly handles occlusion in a distributed fashion, a significant improvement over our earlier tracking results [13]. We conclude with simulations demonstrating our algorithm's robustness to occlusions.

## 2 Geometric Hand Modeling

Structurally, the hand is composed of sixteen approximately rigid components: three phalanges or links for each finger and thumb, as well as the palm [1]. As proposed by [2, 3], we model each rigid body by one or more truncated quadrics (ellipsoids, cones, and cylinders) of fixed size. These geometric primitives are well matched to the true geometry of the hand, allow tracking from arbitrary orientations (in contrast to 2.5D "cardboard" models [5, 9]), and permit efficient computation of projected boundaries and silhouettes [3]. Figure 1 shows the edges and silhouettes corresponding to a sample hand model configuration. Note that only a coarse model of the hand's geometry is necessary for tracking.

### 2.1 Kinematic Representation and Constraints

The kinematic constraints between different hand model components are well described by revolute joints [1]. Figure 2(a) shows a graph describing this kinematic structure, in which nodes correspond to rigid bodies and edges to joints. The two joints connecting the phalanges of each finger and thumb have a single rotational degree of freedom, while the joints connecting the base of each finger to the palm have two degrees of freedom (corresponding to grasping and spreading motions). These twenty angles, combined with the palm's global position and orientation, provide 26 degrees of freedom. Forward kinematic transformations may be used to determine the finger positions corresponding to a given set of joint angles. While most model–based hand trackers use this joint angle parameterization, we instead explore a redundant representation in which the $i^{th}$ rigid body is described by its position $q_i$ and orientation $r_i$ (a unit quaternion). Let $x_i = (q_i, r_i)$ denote this *local* description of each component, and $x = \{x_1, \ldots, x_{16}\}$ the overall hand configuration.

Clearly, there are dependencies among the elements of $x$ implied by the kinematic con-

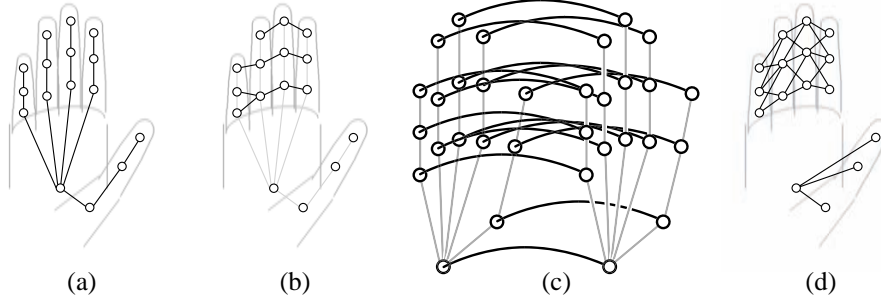

Figure 2: Graphs describing the hand model's constraints. (a) Kinematic constraints ($\mathcal{E}_K$) derived from revolute joints. (b) Structural constraints ($\mathcal{E}_S$) preventing 3D component intersections. (c) Dynamics relating two consecutive time steps. (d) Occlusion consistency constraints ($\mathcal{E}_O$).

straints. Let $\mathcal{E}_K$ be the set of all pairs of rigid bodies which are connected by joints, or equivalently the edges in the kinematic graph of Fig. 2(a). For each joint $(i,j) \in \mathcal{E}_K$, define an indicator function $\psi_{i,j}^K(x_i, x_j)$ which is equal to one if the pair $(x_i, x_j)$ are valid rigid body configurations associated with *some* setting of the angles of joint $(i,j)$, and zero otherwise. Viewing the component configurations $x_i$ as random variables, the following prior explicitly enforces all constraints implied by the original joint angle representation:

$$p_K(x) \propto \prod_{(i,j)\in\mathcal{E}_K} \psi_{i,j}^K(x_i, x_j) \qquad (1)$$

Equation (1) shows that $p_K(x)$ is an undirected graphical model, whose Markov structure is described by the graph representing the hand's kinematic structure (Fig. 2(a)).

## 2.2  Structural and Temporal Constraints

In reality, the hand's joint angles are coupled because different fingers can never occupy the same physical volume. This constraint is complex in a joint angle parameterization, but simple in our local representation: the position and orientation of every pair of rigid bodies must be such that their component quadric surfaces do not intersect.

We approximate this ideal constraint in two ways. First, we only explicitly constrain those pairs of rigid bodies which are most likely to intersect, corresponding to the edges $\mathcal{E}_S$ of the graph in Fig. 2(b). Furthermore, because the relative orientations of each finger's quadrics are implicitly constrained by the kinematic prior $p_K(x)$, we may detect most intersections based on the distance between object centroids. The structural prior is then given by

$$p_S(x) \propto \prod_{(i,j)\in\mathcal{E}_S} \psi_{i,j}^S(x_i, x_j) \qquad \psi_{i,j}^S(x_i, x_j) = \begin{cases} 1 & \|q_i - q_j\| > \delta_{i,j} \\ 0 & \text{otherwise} \end{cases} \qquad (2)$$

where $\delta_{i,j}$ is determined from the quadrics composing rigid bodies $i$ and $j$. Empirically, we find that this constraint helps prevent different fingers from tracking the same image data.

In order to track hand motion, we must model the hand's dynamics. Let $x_i^t$ denote the position and orientation of the $i^{th}$ hand component at time $t$, and $x^t = \{x_1^t, \ldots, x_{16}^t\}$. For each component at time $t$, our dynamical model adds a Gaussian potential connecting it to the corresponding component at the previous time step (see Fig. 2(c)):

$$p_T\left(x^t \mid x^{t-1}\right) = \prod_{i=1}^{16} \mathcal{N}\left(x_i^t - x_i^{t-1}; 0, \Lambda_i\right) \qquad (3)$$

Although this temporal model is factorized, the kinematic constraints at the following time step implicitly couple the corresponding random walks. These dynamics can be justified as the maximum entropy model given observations of the nodes' marginal variances $\Lambda_i$.

## 3 Observation Model

Skin colored pixels have predictable statistics, which we model using a histogram distribution $p_{\text{skin}}$ estimated from training patches [14]. Images without people were used to create a histogram model $p_{\text{bkgd}}$ of non–skin pixels. Let $\Omega(x)$ denote the silhouette of projected hand configuration $x$. Then, assuming pixels $\Upsilon$ are independent, an image $y$ has likelihood

$$p_C(y \mid x) = \prod_{u \in \Omega(x)} p_{\text{skin}}(u) \prod_{v \in \Upsilon \setminus \Omega(x)} p_{\text{bkgd}}(v) \propto \prod_{u \in \Omega(x)} \frac{p_{\text{skin}}(u)}{p_{\text{bkgd}}(u)} \quad (4)$$

The final expression neglects the proportionality constant $\prod_{v \in \Upsilon} p_{\text{bkgd}}(v)$, which is independent of $x$, and thereby limits computation to the silhouette region [8].

### 3.1 Distributed Occlusion Reasoning

In configurations where there is no self–occlusion, $p_C(y \mid x)$ decomposes as a product of local likelihood terms involving the projections $\Omega(x_i)$ of individual hand components [13]. To allow a similar decomposition (and hence distributed inference) when there is occlusion, we augment the configuration $x_i$ of each node with a set of binary hidden variables $z_i = \{z_{i(u)}\}_{u \in \Upsilon}$. Letting $z_{i(u)} = 0$ if pixel $u$ in the projection of rigid body $i$ is occluded by *any* other body, and 1 otherwise, the color likelihood (eq. (4)) may be rewritten as

$$p_C(y \mid x, z) = \prod_{i=1}^{16} \prod_{u \in \Omega(x_i)} \left( \frac{p_{\text{skin}}(u)}{p_{\text{bkgd}}(u)} \right)^{z_{i(u)}} = \prod_{i=1}^{16} p_C(y \mid x_i, z_i) \quad (5)$$

Assuming they are set consistently with the hand configuration $x$, the hidden occlusion variables $z$ ensure that the likelihood of each pixel in $\Omega(x)$ is counted exactly once.

We may enforce consistency of the occlusion variables using the following function:

$$\eta(x_j, z_{i(u)}; x_i) = \begin{cases} 0 & \text{if } x_j \text{ occludes } x_i, u \in \Omega(x_j), \text{ and } z_{i(u)} = 1 \\ 1 & \text{otherwise} \end{cases} \quad (6)$$

Note that because our rigid bodies are convex and nonintersecting, they can never take mutually occluding configurations. The constraint $\eta(x_j, z_{i(u)}; x_i)$ is zero precisely when pixel $u$ in the projection of $x_i$ should be occluded by $x_j$, but $z_{i(u)}$ is in the unoccluded state. The following potential encodes all of the occlusion relationships between nodes $i$ and $j$:

$$\psi_{i,j}^O(x_i, z_i, x_j, z_j) = \prod_{u \in \Upsilon} \eta(x_j, z_{i(u)}; x_i)\, \eta(x_i, z_{j(u)}; x_j) \quad (7)$$

These occlusion constraints exist between all pairs of nodes. As with the structural prior, we enforce only those pairs $\mathcal{E}_O$ (see Fig. 2(d)) most prone to occlusion:

$$p_O(x, z) \propto \prod_{(i,j) \in \mathcal{E}_O} \psi_{i,j}^O(x_i, z_i, x_j, z_j) \quad (8)$$

Figure 3 shows a factor graph for the occlusion relationships between $x_i$ and its neighbors, as well as the observation potential $p_C(y \mid x_i, z_i)$. The occlusion potential $\eta(x_j, z_{i(u)}; x_i)$ has a very weak dependence on $x_i$, depending only on whether $x_i$ is behind $x_j$ relative to the camera.

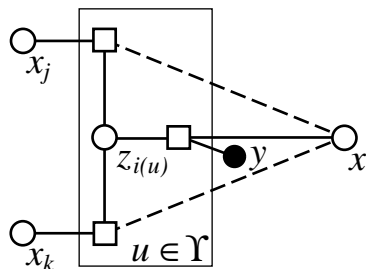

Figure 3: Factor graph showing $p(y \mid x_i, z_i)$, and the occlusion constraints placed on $x_i$ by $x_j, x_k$. Dashed lines denote weak dependencies. The plate is replicated once per pixel.

### 3.2 Modeling Edge Filter Responses

Edges provide another important hand tracking cue. Using boundaries labeled in training images, we estimated a histogram $p_{\text{on}}$ of the response of a derivative of Gaussian filter steered to the edge's orientation [8, 10]. A similar histogram $p_{\text{off}}$ was estimated for filter outputs at

randomly chosen locations. Let $\Pi(x)$ denote the oriented edges in the projection of model configuration $x$. Then, again assuming pixel independence, image $y$ has edge likelihood

$$p_E(y \mid x, z) \propto \prod_{u \in \Pi(x)} \frac{p_{\text{on}}(u)}{p_{\text{off}}(u)} = \prod_{i=1}^{16} \prod_{u \in \Pi(x_i)} \left( \frac{p_{\text{on}}(u)}{p_{\text{off}}(u)} \right)^{z_{i(u)}} = \prod_{i=1}^{16} p_E(y \mid x_i, z_i) \quad (9)$$

where we have used the same occlusion variables $z$ to allow a local decomposition.

## 4 Nonparametric Belief Propagation

Over the previous sections, we have shown that a redundant, local representation of the geometric hand model's configuration $x^t$ allows $p\left(x^t \mid y^t\right)$, the posterior distribution of the hand model at time $t$ given image observations $y^t$, to be written as

$$p\left(x^t \mid y^t\right) \propto \sum_{z^t} p_K(x^t) p_S(x^t) p_O(x^t, z^t) \left[ \prod_{i=1}^{16} p_C(y^t \mid x_i^t, z_i^t) p_E(y^t \mid x_i^t, z_i^t) \right] \quad (10)$$

The summation marginalizes over the hidden occlusion variables $z^t$, which were needed to locally decompose the edge and color likelihoods. When $\tau$ video frames are observed, the overall posterior distribution is given by

$$p\left(x \mid y\right) \propto \prod_{t=1}^{\tau} p\left(x^t \mid y^t\right) p_T(x^t \mid x^{t-1}) \quad (11)$$

Excluding the potentials involving occlusion variables, which we discuss in detail in Sec. 4.2, eq. (11) is an example of a pairwise Markov random field:

$$p\left(x \mid y\right) \propto \prod_{(i,j) \in \mathcal{E}} \psi_{i,j}\left(x_i, x_j\right) \prod_{i \in \mathcal{V}} \psi_i\left(x_i, y\right) \quad (12)$$

Hand tracking can thus be posed as inference in a graphical model, a problem we propose to solve using *belief propagation* (BP) [15]. At each BP iteration, some node $i \in \mathcal{V}$ calculates a message $m_{ij}\left(x_j\right)$ to be sent to a neighbor $j \in \Gamma(i) \triangleq \{j \mid (i,j) \in \mathcal{E}\}$:

$$m_{ij}^n\left(x_j\right) \propto \int_{x_i} \psi_{j,i}\left(x_j, x_i\right) \psi_i\left(x_i, y\right) \prod_{k \in \Gamma(i) \backslash j} m_{ki}^{n-1}\left(x_i\right) \, dx_i \quad (13)$$

At any iteration, each node can produce an approximation $\hat{p}(x_i \mid y)$ to the marginal distribution $p\left(x_i \mid y\right)$ by combining the incoming messages with the local observation:

$$\hat{p}^n(x_i \mid y) \propto \psi_i\left(x_i, y_i\right) \prod_{j \in \Gamma(i)} m_{ji}^n\left(x_i\right) \quad (14)$$

For tree–structured graphs, the *beliefs* $\hat{p}^n(x_i \mid y)$ will converge to the true marginals $p\left(x_i \mid y\right)$. On graphs with cycles, BP is approximate but often highly accurate [15].

### 4.1 Nonparametric Representations

For the hand tracking problem, the rigid body configurations $x_i$ are six–dimensional continuous variables, making accurate discretization intractable. Instead, we employ nonparametric, particle–based approximations to these messages using the nonparametric belief propagation (NBP) algorithm [11, 12]. In NBP, each message is represented using either a sample–based density estimate (a mixture of Gaussians) or an analytic function. Both types of messages are needed for hand tracking, as we discuss below. Each NBP message update involves two stages: sampling from the estimated marginal, followed by Monte Carlo approximation of the outgoing message. For the general form of these updates, see [11]; the following sections focus on the details of the hand tracking implementation.

The hand tracking application is complicated by the fact that the orientation component $r_i$ of $x_i = (q_i, r_i)$ is an element of the rotation group $SO(3)$. Following [10], we represent

orientations as unit quaternions, and use a linearized approximation when constructing density estimates, projecting samples back to the unit sphere as necessary. This approximation is most appropriate for densities with tightly concentrated rotational components.

## 4.2 Marginal Computation

BP's estimate of the belief $\hat{p}(x_i \mid y)$ is equal to the product of the incoming messages from neighboring nodes with the local observation potential (see eq. (14)). NBP approximates this product using importance sampling, as detailed in [13] for cases where there is no self–occlusion. First, $M$ samples are drawn from the product of the incoming kinematic and temporal messages, which are Gaussian mixtures. We use a recently proposed multi-scale Gibbs sampler [16] to efficiently draw accurate (albeit approximate) samples, while avoiding the exponential cost associated with direct sampling (a product of $d$ $M$–Gaussian mixtures contains $M^d$ Gaussians). Following normalization of the rotational component, each sample is assigned a weight equal to the product of the color and edge likelihoods with any structural messages. Finally, the computationally efficient "rule of thumb" heuristic [17] is used to set the bandwidth of Gaussian kernels placed around each sample.

To derive BP updates for the occlusion masks $z_i$, we first cluster $(x_i, z_i)$ for each hand component so that $p(x^t, z^t \mid y^t)$ has a pairwise form (as in eq. (12)). In principle, NBP could manage occlusion constraints by sampling candidate occlusion masks $z_i$ along with rigid body configurations $x_i$. However, due to the exponentially large number of possible occlusion masks, we employ a more efficient analytic approximation.

Consider the BP message sent from $x_j$ to $(z_i, x_i)$, calculated by applying eq. (13) to the occlusion potential $\prod_u \eta(x_j, z_{i(u)}; x_i)$. We assume that $\hat{p}(x_j \mid y)$ is well separated from any candidate $x_i$, a situation typically ensured by the kinematic and structural constraints. The occlusion constraint's weak dependence on $x_i$ (see Fig. 3) then separates the message computation into two cases. If $x_i$ lies in front of typical $x_j$ configurations, the BP message $\mu_{j,i(u)}(z_{i(u)})$ is uninformative. If $x_i$ is occluded, the message approximately equals

$$\mu_{j,i(u)}(z_{i(u)} = 0) = 1 \qquad \mu_{j,i(u)}(z_{i(u)} = 1) = 1 - \Pr\left[u \in \Omega(x_j)\right] \qquad (15)$$

where we have neglected correlations among pixel occlusion states, and where the probability is computed with respect to $\hat{p}(x_j \mid y)$. By taking the product of these messages $\mu_{k,i(u)}(z_{i(u)})$ from all potential occluders $x_k$ and normalizing, we may determine an approximation to the marginal occlusion probability $\nu_{i(u)} \triangleq \Pr[z_{i(u)} = 0]$.

Because the color likelihood $p_C(y \mid x_i, z_i)$ factorizes across pixels $u$, the BP approximation to $p_C(y \mid x_i)$ may be written in terms of these marginal occlusion probabilites:

$$p_C(y \mid x_i) \propto \prod_{u \in \Omega(x_i)} \left[ \nu_{i(u)} + (1 - \nu_{i(u)}) \left( \frac{p_{\text{skin}}(u)}{p_{\text{bkgd}}(u)} \right) \right] \qquad (16)$$

Intuitively, this equation downweights the color evidence at pixel $u$ as the probability of that pixel's occlusion increases. The edge likelihood $p_E(y \mid x_i)$ averages over $z_i$ similarly. The NBP estimate of $\hat{p}(x_i \mid y)$ is determined by sampling configurations of $x_i$ as before, and reweighting them using these occlusion–sensitive likelihood functions.

## 4.3 Message Propagation

To derive the propagation rule for non–occlusion edges, as suggested by [18] we rewrite the message update equation (13) in terms of the marginal distribution $\hat{p}(x_i \mid y)$:

$$m_{ij}^n(x_j) = \alpha \int_{x_i} \psi_{j,i}(x_j, x_i) \frac{\hat{p}^{n-1}(x_i \mid y)}{m_{ji}^{n-1}(x_i)} \, dx_i \qquad (17)$$

Our explicit use of the current marginal estimate $\hat{p}^{n-1}(x_i \mid y)$ helps focus the Monte Carlo approximation on the most important regions of the state space. Note that messages sent

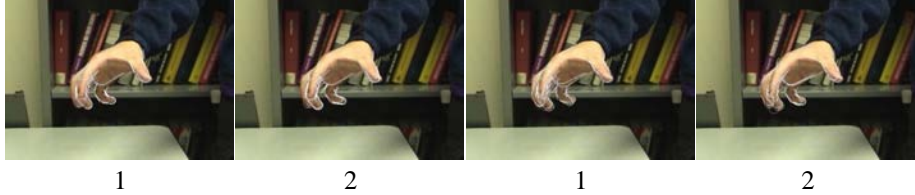

|    1    |    2    |    1    |    2    |

Figure 4: Refinement of a coarse initialization following one and two NBP iterations, both without (left) and with (right) occlusion reasoning. Each plot shows the projection of the five most significant modes of the estimated marginal distributions. Note the difference in middle finger estimates.

along kinematic, structural, and temporal edges depend only on the belief $\hat{p}(x_i \mid y)$ following marginalization over occlusion variables $z_i$.

Details and pseudocode for the message propagation step are provided in [13]. For kinematic constraints, we sample uniformly among permissable joint angles, and then use forward kinematics to propagate samples from $\hat{p}^{n-1}(x_i \mid y)/m_{ji}^{n-1}(x_i)$ to hypothesized configurations of $x_j$. Following [12], temporal messages are determined by adjusting the bandwidths of the current marginal estimate $\hat{p}(x_i \mid y)$ to match the temporal covariance $\Lambda_i$. Because structural potentials (eq. (2)) equal one for all state configurations outside some ball, the ideal structural messages are not finitely integrable. We therefore approximate the structural message $m_{ij}(x_j)$ as an analytic function equal to the weights of all kernels in $\hat{p}(x_i \mid y)$ outside a ball centered at $q_j$, the position of $x_j$.

## 5  Simulations

We now present a set of computational examples which investigate the performance of our distributed occlusion reasoning; see [13] for additional simulations. In Fig. 4, we use NBP to refine a coarse, user–supplied initialization into an accurate estimate of the hand's configuration in a single image. When occlusion constraints are neglected, the NBP estimates associate the ring and middle fingers with the same image pixels, and miss the true middle finger location. With proper occlusion reasoning, however, the correct hand configuration is identified. Using $M = 200$ particles, our Matlab implementation requires about one minute for each NBP iteration (an update of all messages in the graph).

Video sequences demonstrating the NBP hand tracker are available at `http://ssg.mit.edu/nbp/`. Selected frames from two of these sequences are shown in Fig. 5, in which filtered estimates are computed by a single "forward" sequence of temporal message updates. The initial frame was approximately initialized manually. The first sequence shows successful tracking through a partial occlusion of the ring finger by the middle finger, while the second shows a grasping motion in which the fingers occlude each other. For both of these sequences, rough tracking (not shown) is possible without occlusion reasoning, since all fingers are the same color and the background is unambiguous. However, we find that stability improves when occlusion reasoning is used to properly discount obscured edges and silhouettes.

## 6  Discussion

Sigal et. al. [10] developed a three–dimensional NBP person tracker which models the conditional distribution of each linkage's location, given its neighbor, via a Gaussian mixture estimated from training data. In contrast, we have shown that an NBP tracker may be built around the local structure of the true kinematic constraints. Conceptually, this has the advantage of providing a clearly specified, globally consistent generative model whose properties can be analyzed. Practically, our formulation avoids the need to explicitly approximate kinematic constraints, and allows us to build a functional tracker without the need for precise, labelled training data.

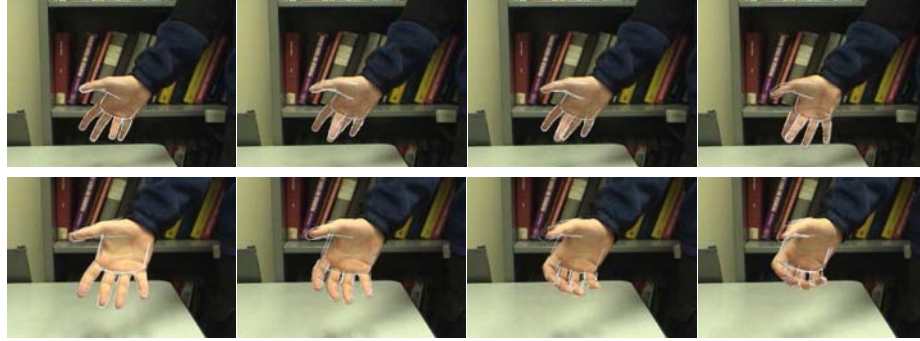

Figure 5: Four frames from two different video sequences: a hand rotation containing finger occlusion (top), and a grasping motion (bottom). We show the projections of NBP's marginal estimates.

We have described the graphical structure underlying a kinematic model of the hand, and used this model to build a tracking algorithm using nonparametric BP. By appropriately augmenting the model's state, we are able to perform occlusion reasoning in a distributed fashion. The modular state representation and robust, local computations of NBP offer a solution particularly well suited to visual tracking of articulated objects.

### Acknowledgments

The authors thank C. Mario Christoudias and Michael Siracusa for their help with video data collection, and Michael Black, Alexander Ihler, Michael Isard, and Leonid Sigal for helpful conversations. This research was supported in part by DARPA Contract No. NBCHD030010.

### References

[1] Y. Wu and T. S. Huang. Hand modeling, analysis, and recognition. *IEEE Signal Proc. Mag.*, pages 51–60, May 2001.

[2] J. M. Rehg and T. Kanade. DigitEyes: Vision–based hand tracking for human–computer interaction. In *Proc. IEEE Workshop on Non–Rigid and Articulated Objects*, 1994.

[3] B. Stenger, P. R. S. Mendonca, and R. Cipolla. Model–based 3D tracking of an articulated hand. In *CVPR*, volume 2, pages 310–315, 2001.

[4] J. MacCormick and M. Isard. Partitioned sampling, articulated objects, and interface–quality hand tracking. In *ECCV*, volume 2, pages 3–19, 2000.

[5] Y. Wu, J. Y. Lin, and T. S. Huang. Capturing natural hand articulation. In *ICCV*, 2001.

[6] B. Stenger, A. Thayananthan, P. H. S. Torr, and R. Cipolla. Filtering using a tree–based estimator. In *ICCV*, pages 1063–1070, 2003.

[7] D. Ramanan and D. A. Forsyth. Finding and tracking people from the bottom up. In *CVPR*, volume 2, pages 467–474, 2003.

[8] J. M. Coughlan and S. J. Ferreira. Finding deformable shapes using loopy belief propagation. In *ECCV*, volume 3, pages 453–468, 2002.

[9] Y. Wu, G. Hua, and T. Yu. Tracking articulated body by dynamic Markov network. In *ICCV*, pages 1094–1101, 2003.

[10] L. Sigal, M. Isard, B. H. Sigelman, and M. J. Black. Attractive people: Assembling loose–limbed models using nonparametric belief propagation. In *NIPS*, 2003.

[11] E. B. Sudderth, A. T. Ihler, W. T. Freeman, and A. S. Willsky. Nonparametric belief propagation. In *CVPR*, volume 1, pages 605–612, 2003.

[12] M. Isard. PAMPAS: Real–valued graphical models for computer vision. In *CVPR*, volume 1, pages 613–620, 2003.

[13] Erik B. Sudderth, M. I. Mandel, W. T. Freeman, and A. S. Willsky. Visual hand tracking using nonparametric belief propagation. MIT LIDS TR2603, May 2004. Presented at CVPR Workshop on Generative Model Based Vision, June 2004. http://ssg.mit.edu/nbp/.

[14] M. J. Jones and J. M. Rehg. Statistical color models with application to skin detection. *IJCV*, 46(1):81–96, 2002.

[15] J. S. Yedidia, W. T. Freeman, and Y. Weiss. Constructing free energy approximations and generalized belief propagation algorithms. Technical Report 2004-040, MERL, May 2004.

[16] A. T. Ihler, E. B. Sudderth, W. T. Freeman, and A. S. Willsky. Efficient multiscale sampling from products of Gaussian mixtures. In *NIPS*, 2003.

[17] B. W. Silverman. *Density Estimation for Statistics and Data Analysis*. Chapman & Hall, 1986.

[18] D. Koller, U. Lerner, and D. Angelov. A general algorithm for approximate inference and its application to hybrid Bayes nets. In *UAI 15*, pages 324–333, 1999.
